# Cholinergic suppression of transmission may allow combined associative memory function and self-organization in the neocortex.

**Michael E. Hasselmo and Milos Cekic**
Department of Psychology and Program in Neurosciences,
Harvard University, 33 Kirkland St., Cambridge, MA 02138
hasselmo@katla.harvard.edu

## Abstract

Selective suppression of transmission at feedback synapses during learning is proposed as a mechanism for combining associative feedback with self-organization of feedforward synapses. Experimental data demonstrates cholinergic suppression of synaptic transmission in layer I (feedback synapses), and a lack of suppression in layer IV (feedforward synapses). A network with this feature uses local rules to learn mappings which are not linearly separable. During learning, sensory stimuli and desired response are simultaneously presented as input. Feedforward connections form self-organized representations of input, while suppressed feedback connections learn the transpose of feedforward connectivity. During recall, suppression is removed, sensory input activates the self-organized representation, and activity generates the learned response.

## 1    INTRODUCTION

The synaptic connections in most models of the cortex can be defined as either associative or self-organizing on the basis of a single feature: the relative influence of modifiable synapses on post-synaptic activity during learning (figure 1). In associative memories, post-synaptic activity during learning is determined by nonmodifiable afferent input connections, with no change in the storage due to synaptic transmission at modifiable synapses (Anderson, 1983; McNaughton and Morris, 1987). In self-organization, post-synaptic activity is predominantly influenced by the modifiable synapses, such that modification of synapses influences subsequent learning (Von der Malsburg, 1973; Miller et al., 1990). Models of cortical function must combine the capacity to form new representations and store associations between these representations. Networks combining self-organization and associative memory function can learn complex mapping functions with more biologically plausible learning rules (Hecht-Nielsen, 1987; Carpenter et al., 1991; Dayan et al.,

1995), but must control the influence of feedback associative connections on self-organization. Some networks use special activation dynamics which prevent feedback from influencing activity unless it coincides with feedforward activity (Carpenter et al., 1991). A new network alternately shuts off feedforward and feedback synaptic transmission (Dayan et al., 1995).

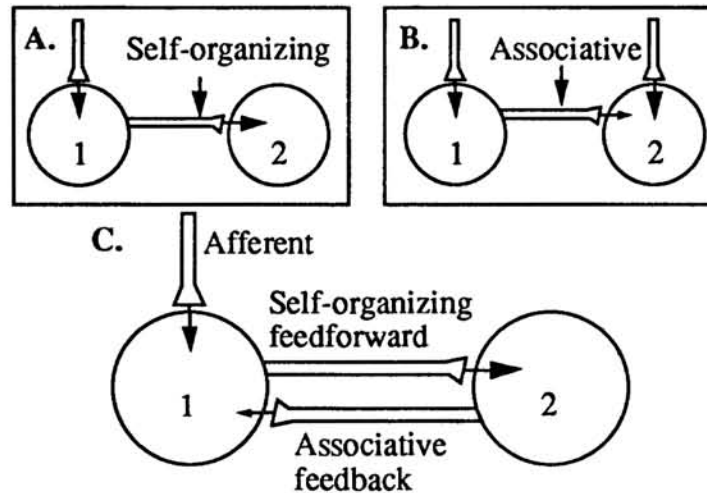

**Figure 1** - Defining characteristics of self-organization and associative memory. A. At self-organizing synapses, post-synaptic activity during learning depends predominantly upon transmission at the modifiable synapses. B. At synapses mediating associative memory function, post-synaptic activity during learning does not depend primarily on the modifiable synapses, but is predominantly influenced by separate afferent input. C. Self-organization and associative memory function can be combined if associative feedback synapses are selectively suppressed during learning but not recall.

Here we present a model using selective suppression of feedback synaptic transmission during learning to allow simultaneous self-organization and association between two regions. Previous experiments show that the neuromodulator acetylcholine selectively suppresses synaptic transmission within the olfactory cortex (Hasselmo and Bower, 1992; 1993) and hippocampus (Hasselmo and Schnell, 1994). If the model is valid for neocortical structures, cholinergic suppression should be stronger for feedback but not feedforward synapses. Here we review experimental data (Hasselmo and Cekic, 1996) comparing cholinergic suppression of synaptic transmission in layers with predominantly feedforward or feedback synapses.

## 2. BRAIN SLICE PHYSIOLOGY

As shown in Figure 2, we utilized brain slice preparations of the rat somatosensory neocortex to investigate whether cholinergic suppression of synaptic transmission is selective for feedback but not feedforward synaptic connections. This was possible because feedforward and feedback connections show different patterns of termination in neocortex. As shown in Figure 2, Layer I contains primarily feedback synapses from other cortical regions (Cauller and Connors, 1994), whereas layer IV contains primarily afferent synapses from the thalamus and feedforward synapses from more primary neocortical structures (Van Essen and Maunsell, 1983). Using previously developed techniques (Cauller and Connors, 1994; Li and Cauller, 1995) for testing of the predominantly feedback connections in layer I, we stimulated layer I and recorded in layer I (a cut prevented spread of

activity from layers II and III). For testing the predominantly feedforward connections terminating in layer IV, we elicited synaptic potentials by stimulating the white matter deep to layer VI and recorded in layer IV. We tested suppression by measuring the change in height of synaptic potentials during perfusion of the cholinergic agonist carbachol at 100µM. Figure 3 shows that perfusion of carbachol caused much stronger suppression of synaptic transmission in layer I as compared to layer IV (Hasselmo and Cekic, 1996), suggesting that cholinergic suppression of transmission is selective for feedback synapses and not for feedforward synapses.

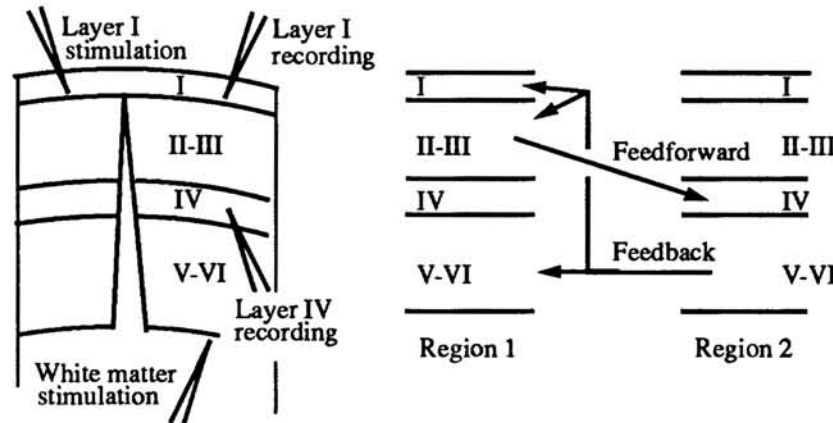

Figure 2. A. Brain slice preparation of somatosensory cortex showing location of stimulation and recording electrodes for testing suppression of synaptic transmission in layer I and in layer IV. Experiment based on procedures developed by Cauller (Cauller and Connors, 1994; Li and Cauller, 1995). B. Anatomical pattern of feedforward and feedback connectivity within cortical structures (based on Van Essen and Maunsell, 1983).

## Feedforward - layer IV

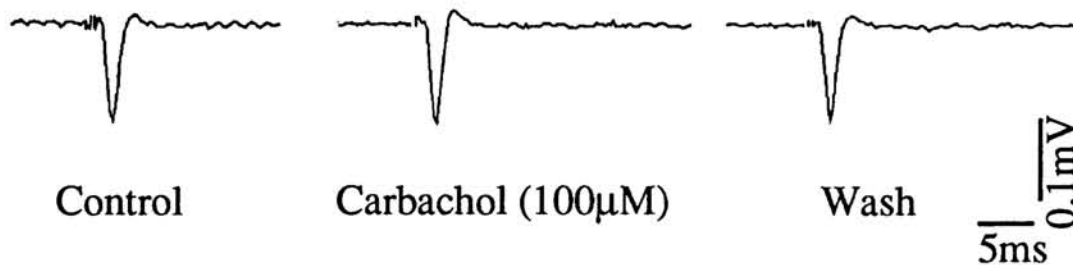

Control          Carbachol (100µM)          Wash

## Feedback - layer I

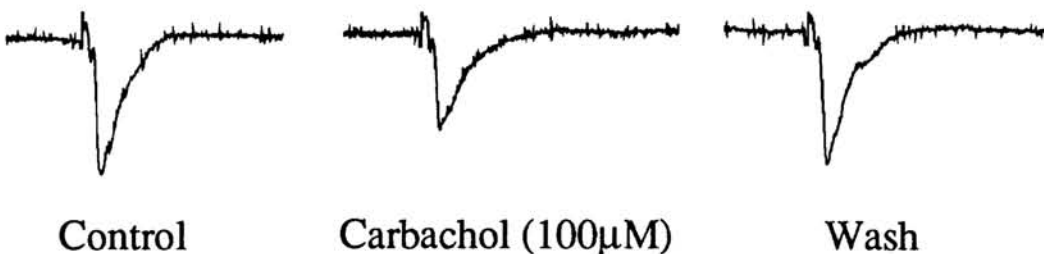

Control          Carbachol (100µM)          Wash

Figure 3 - Suppression of transmission in somatosensory neocortex. Top: Synaptic potentials recorded in layer IV (where feedforward and afferent synapses predominate) show little effect of 100µM carbachol. Bottom: Synaptic potentials recorded in layer I (where feedback synapses predominate) show suppression in the presence of 100µM carbachol.

## 3. COMPUTATIONAL MODELING

These experimental results supported the use of selective suppression in a computational model (Hasselmo and Cekic, 1996) with self-organization in its feedforward synaptic connections and associative memory function in its feedback synaptic connections (Figs 1 and 4). The proposed network uses local, Hebb-type learning rules supported by evidence on the physiology of long-term potentiation in the hippocampus (Gustafsson and Wigstrom, 1986). The learning rule for each set of connections in the network takes the form:

$$\Delta W_{ij}^{(x, y)} = \eta \, (a_i^{(y)} - \theta^{(y)}) \, g \, (a_j^{(x)})$$

Where $W^{(x, y)}$ designates the connections from region x to region y, $\theta$ is the threshold of synaptic modification in region y, $\eta$ is the rate of modification, and the output function is $g(a_i^{(x)}) = [\tanh(a_j^{(x)} - \mu^{(x)})]_+$ where $[]_+$ represents the constraint to positive values only. Feedforward connections ($W_{ij}^{(x<y)}$) have self-organizing properties, while feedback connections ($W_{ij}^{(x>=y)}$) have associative memory properties. This difference depends entirely upon the selective suppression of feedback synapses during learning, which is implemented in the activation rule in the form (1-c). For the entire network, the activation rule takes the form:

$$a_i^{(y)} = A_i^{(y)} + \sum_{x=1}^{M} \sum_{k=1}^{n(x)} W_{ik}^{(x<y)} g(a_k^{(x)}) + \sum_{x=1}^{N} \sum_{k=1}^{n(x)} (1-c) \, W_{ik}^{(x \geq y)} g(a_k^{(x)}) - \sum_{k=1}^{n(y)} H_{ik}^{(y)} (g(a_k^{(y)}))$$

where $a_i^{(y)}$ represents the activity of each of the $n^{(y)}$ neurons in region y, $a_k^{(x)}$ is the activity of each of the $n^{(x)}$ neurons in other regions x, M is the total number of regions providing feedforward input, N is the total number of regions providing feedback input , $A_i^{(y)}$ is the input pattern to region y, $H^{(y)}$ represents the inhibition between neurons in region y, and (1 - c) represents the suppression of synaptic transmission. During learning, c takes a value between 0 and 1. During recall, suppression is removed, c = 0. In this network, synapses (W) between regions only take positive values, reflecting the fact that long-range connections between cortical regions consist of excitatory synapses arising from pyramidal cells. Thus, inhibition mediated by the local inhibitory interneurons within a region is represented by a separate inhibitory connectivity matrix H.

After each step of learning, the total weight of synaptic connections is normalized pre-synaptically for each neuron j in each region:

$$W_{ij}(t+1) = [W_{ij}(t) + \Delta W_{ij}(t)] / \left( \sqrt{\sum_{i=1}^{n} [W_{ij}(t) + \Delta W_{ij}(t)]^2} \right)$$

Synaptic weights are then normalized post-synaptically for each neuron i in each region (replacing i with j in the sum in the denominator in equation 3). This normalization of synaptic strength represents slower cellular mechanisms which redistribute pre and post-synaptic resources for maintaining synapses depending upon local influences.

In these simulations, both the sensory input stimuli and the desired output response to be learned are presented as afferent input to the neurons in region 1. Most networks using error-based learning rules consist of feedforward architectures with separate layers of input and output units. One can imagine this network as an auto-encoder network folded back on itself, with both input and output units in region 1, and hidden units in region 2.

As an example of its functional properties, the network presented here was trained on the XOR problem. The XOR problem has previously been used as an example of the capability of error based training schemes for solving problems which are not linearly separable. The specific characteristics of the network and patterns used for this simulation are shown in figure 4. The two logical states of each component of the XOR problem are represented by two separate units (designated on or off in figures 4 and 5), ensuring that activation of the network is equal for each input condition. The problem has the appearance of two XOR problems with inverse logical states being solved simultaneously.

As shown in figure 4, the input and desired output of the network are presented simultaneously during learning to region 1. The six neurons in region 1 project along feedforward connections to four neurons in region 2, the hidden units of the network. These four neurons project along feedback connections to the six neurons in region 1. All connections take random initial weights. During learning, the feedforward connections undergo self-organization which ultimately causes the hidden units to become feature detectors responding to each of the four patterns of input to region 1. Thus, the rows of the feedforward synaptic connectivity matrix gradually take the form of the individual input patterns.

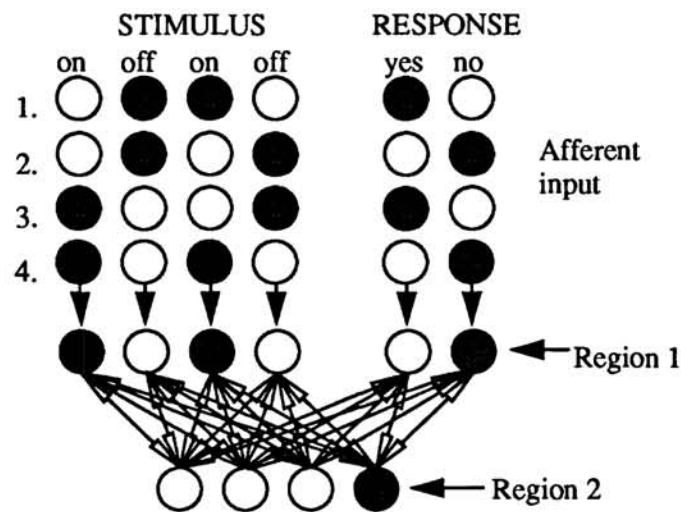

Figure 4 - Network for learning the XOR problem, with 6 units in region 1 and 4 units in region 2. Four different patterns of afferent input are presented successively to region 1. The input stimuli of the XOR problem are represented by the four units on the left, and the desired output designation of XOR or not-XOR is represented by the two units on the right. The XOR problem has four basic states: on-off and off-on on the input is categorized by yes on the output, while on-on and off-off on the input is categorized by no on the output.

Modulation is applied during learning in the form of selective suppression of synaptic transmission along feedback connections (this suppression need not be complete), giving these connections associative memory function. Hebbian synaptic modification causes these connections to link each of the feature detecting hidden units in region 2 with the cells in region 1 activated by the pattern to which the hidden unit responds. Gradually, the feedback synaptic connectivity matrix becomes the transpose of the feedforward connectivity matrix. (Parameters used in simulation: Aj(1) = 0 or 1, h = 2.0, q(1) = 0.5, q(2) = 0.6, (1) = 0.2, (2) = 0.5, c = 1.0 and Hik(2) = 0.6). Function was similar and convergence was obtained more rapidly with c = 0.5. Feedback synaptic transmission prevented con-

vergence during learning when c = 0.367).

During recall, modulation of synaptic transmission is removed, and the various input stim-uli of the XOR problem are presented to region 1 without the corresponding output pat-tern. Activity spreads along the self-organized feedforward connections to activate the specific hidden layer unit responding to that pattern. Activity then spreads back along feedback connections from that particular unit to activate the desired output units. The activity in the two regions settles into a final pattern of recall. Figure 5 shows the settled recall of the network at different stages of learning. It can be seen that the network ini-tially may show little recall activity, or erroneous recall activity, but after several cycles of learning, the network settles into the proper response to each of the XOR problem states. Convergence during learning and recall have been obtained with other problems, includ-ing recognition of whether on units were on the left or right, symmetry of on units, and number of on units. In addition, larger scale problems involving multiple feedforward and feedback layers have been shown to converge.

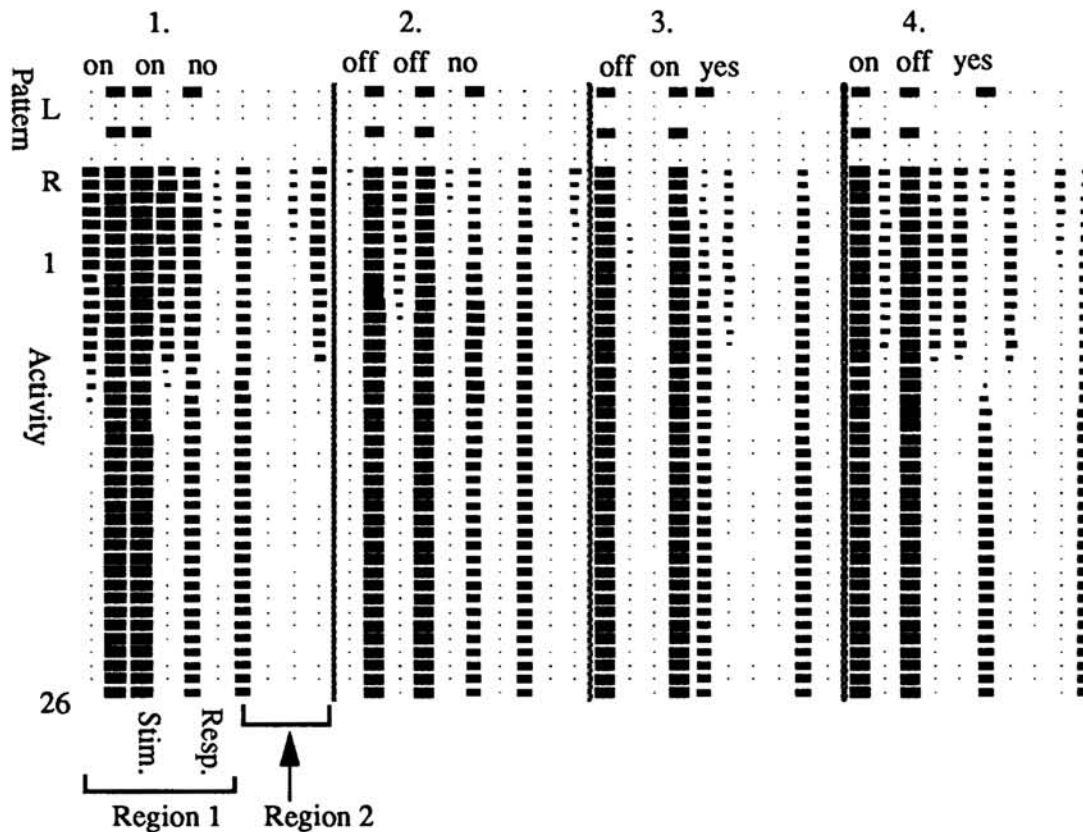

Figure 5 - Output neuronal activity in the network shown at different learning steps. The four input patterns are shown at top. Below these are degraded patterns presented during recall, missing the response components of the input pattern. The output of the 6 region 1 units and the 4 region 2 units are shown at each stage of learning. As learning progresses, gradually one region 2 unit starts to respond selectively to each input pattern, and the cor-rect output unit becomes active in response to the degraded input. Note that as learning progresses the response to pattern 4 changes gradually from incorrect (yes) to correct (no).

**References**

Anderson, J.A. (1983) Cognitive and psychological computation with neural models. IEEE Trans. Systems, Man, Cybern. SMC-13, 799-815.

Carpenter, G.A., Grossberg, S. and Reynolds, J.H. (1991) ARTMAP: Supervised real-time learning and classification of nonstationary data by a self-organizing neural network. Neural Networks 4: 565-588.

Cauller, L.J. and Connors, B.W. (1994) Synaptic physiology of horizontal afferents to layer I in slices of rat SI neocortex. J. Neurosci. 14: 751-762.

Dayan, P., Hinton, G.E., Neal, R.M. and Zemel, R.S. (1995) The Helmholtz machine. Neural Computation .

Gustafsson, B. and Wigstrom, H. (1988) Physiological mechanisms underlying long-term potentiation. Trends Neurosci. 11: 156-162.

Hasselmo, M.E. (1993) Acetylcholine and learning in a cortical associative memory. Neural Computation. 5(1): 32-44.

Hasselmo M.E. and Bower J.M. (1992) Cholinergic suppression specific to intrinsic not afferent fiber synapses in rat piriform (olfactory) cortex. J. Neurophysiol. 67: 1222-1229.

Hasselmo, M.E. and Bower, J.M. (1993) Acetylcholine and Memory. Trends Neurosci. 26: 218-222.

Hasselmo, M.E. and Cekic, M. (1996) Suppression of synaptic transmission may allow combination of associative feedback and self-organizing feedforward connections in the neocortex. Behav. Brain Res. in press.

Hasselmo M.E., Anderson B.P. and Bower J.M. (1992) Cholinergic modulation of cortical associative memory function. J. Neurophysiol. 67: 1230-1246.

Hasselmo M.E. and Schnell, E. (1994) Laminar selectivity of the cholinergic suppression of synaptic transmission in rat hippocampal region CA1: Computational modeling and brain slice physiology. J. Neurosci. 15: 3898-3914.

Hecht-Nielsen, R. (1987) Counterpropagation networks. Applied Optics 26: 4979-4984.

Li, H. and Cauller, L.J. (1995) Acetylcholine modulation of excitatory synaptic inputs from layer I to the superficial layers of rat somatosensory neocortex in vitro. Soc. Neurosci. Abstr. 21: 68.

Linsker, R. (1988) Self-organization in a perceptual network. Computer 21: 105-117.

McNaughton B.L. and Morris R.G.M. (1987) Hippocampal synaptic enhancement and information storage within a distributed memory system. Trends in Neurosci. 10:408-415.

Miller, K.D., Keller, J.B. and Stryker, M.P. (1989) Ocular dominance column development: Analysis and simulation. Science 245: 605-615.

van Essen, D.C. and Maunsell, J.H.R. (1983) Heirarchical organization and functional streams in the visual cortex. Trends Neurosci. 6: 370-375.

von der Malsburg, C. (1973) Self-organization of orientation sensitive cells in the striate cortex. Kybernetik 14: 85-100.